# Scaling Laws in Natural Scenes and the Inference of 3D Shape

**Brian Potetz**
Department of Computer Science
Center for the Neural Basis of Cognition
Carnegie Mellon University
Pittsburgh, PA 15213
bpotetz@cs.cmu.edu

**Tai Sing Lee**
Department of Computer Science
Center for the Neural Basis of Cognition
Carnegie Mellon University
Pittsburgh, PA 15213
tai@cnbc.cmu.edu

## Abstract

This paper explores the statistical relationship between natural images and their underlying range (depth) images. We look at how this relationship changes over scale, and how this information can be used to enhance low resolution range data using a full resolution intensity image. Based on our findings, we propose an extension to an existing technique known as shape recipes [3], and the success of the two methods are compared using images and laser scans of real scenes. Our extension is shown to provide a two-fold improvement over the current method. Furthermore, we demonstrate that ideal linear shape-from-shading filters, when learned from natural scenes, may derive even more strength from shadow cues than from the traditional linear-Lambertian shading cues.

## 1 Introduction

The inference of depth information from single images is typically performed by devising models of image formation based on the physics of light interaction and then inverting these models to solve for depth. Once inverted, these models are highly underconstrained, requiring many assumptions such as Lambertian surface reflectance, smoothness of surfaces, uniform albedo, or lack of cast shadows. Little is known about the relative merits of these assumptions in real scenes. A statistical understanding of the joint distribution of real images and their underlying 3D structure would allow us to replace these assumptions and simplifications with probabilistic priors based on real scenes. Furthermore, statistical studies may uncover entirely new sources of information that are not obvious from physical models. Real scenes are affected by many regularities in the environment, such as the natural geometry of objects, the arrangements of objects in space, natural distributions of light, and regularities in the position of the observer. Few current shape inference algorithms make use of these trends. Despite the potential usefulness of statistical models and the growing success of statistical methods in vision, few studies have been made into the statistical relationship between images and range (depth) images. Those studies that have examined this relationship in nature have uncovered meaningful and exploitable statistical trends in real scenes which may be useful for designing new algorithms in surface inference, and also for understanding how humans perceive depth in real scenes [6, 4, 8].

In this paper, we explore some of the properties of the statistical relationship between images and their underlying range (depth) images in real scenes, using images acquired by laser scanner in natural environments. Specifically, we will examine the cross-covariance between images and range images, and how this structure changes over scale. We then illustrate how our statistical findings can be applied to inference problems by analyzing and extending the shape recipe depth inference algorithm.

## 2 Shape recipes

We will motivate our statistical study with an application. Often, we may have a high-resolution color image of a scene, but only a low spatial resolution range image (range images record the 3D distance between the scene and the camera for each pixel). This often happens if our range image was acquired by applying a stereo depth inference algorithm. Stereo algorithms rely on smoothness constraints, either explicitly or implicitly, and so the high-frequency components of the resulting range image are not reliable [1, 7]. Low-resolution range data may also be the output of a laser range scanner, if the range scanner is inexpensive, or if the scan must be acquired quickly (range scanners typically acquire each pixel sequentially, taking up to several minutes for a high-resolution scan).

It should be possible to improve our estimate of the high spatial frequencies of the range image by using monocular cues from the high-resolution intensity (or color) image. Shape recipes [3, 9] provide one way of doing this. The basic principle of shape recipes is that a relationship between shape and light intensity could be *learned* from the low resolution image pair, and then *extrapolated* and applied to the high resolution intensity image to infer the high spatial frequencies of the range image. One advantage of this approach is that hidden variables important to inference from monocular cues, such as illumination direction and material reflectance properties, might be implicitly learned from the low-resolution range and intensity images. However, for this approach to work, we require some model of how the relationship between shape and intensity changes over scale, which we discuss below.

For shape recipes, both the high resolution intensity image and the low resolution range image are decomposed into steerable wavelet filter pyramids, linearly breaking the image down according to scale and orientation [2]. Linear regression is then used between the highest frequency band of the available low-resolution range image and the corresponding band of the intensity image, to learn a linear filter that best predicts the range band from the image band. The hypothesis of the model is that this filter can then be used to predict high frequency range bands from the high frequency image bands. We describe the implementation in more detail below.

Let $i_{m,\phi}$ and $z_{m,\phi}$ be steerable filter pyramid subbands of the intensity and range image respectively, at spatial resolution $m$ and orientation $\phi$ (both are integers). Number the band levels so that $m=0$ is the highest frequency subband of the intensity image, and $m=n$ is the highest available frequency subband of the low-resolution range image. Thus, higher level numbers correspond to lower spatial frequencies. Shape recipes work by learning a linear filter $k_{n,\phi}$ at level $n$ by minimizing sum-squared error $\sum(z_{n,\phi} - k_{n,\phi} \star i_{n,\phi})^2$, where $\star$ denotes convolution. Higher resolution subbands of the range image are inferred by:

$$\hat{z}_{m,\phi} = \frac{1}{c^{n-m}}(k_{n,\phi} \star i_{m,\phi}) \qquad (1)$$

where $c = 2$. The choice of $c = 2$ in the shape recipe model is motivated by the linear Lambertian shading model [9]. We will discuss this choice of constant in section 3.

The underlying assumption of shape recipes is that the convolution kernel $k_{m,\phi}$ should be roughly constant over the four highest resolution bands of the steerable filter pyramid. This

is based on the idea that shape recipe kernels should vary slowly over scale. In this section, we show mathematically that this model is internally inconsistent. To do this, we first re-express the shape recipe process in the Fourier domain. The operations of shape recipes (pyramid decomposition, convolution, and image reconstruction) are all linear operations, and so they can be combined into a single linear convolution. In other words, we can think of shape recipes as inferring the high resolution range data $z_{high}$ via a single convolution

$$Z_{high}(u, v) = I(u, v) \cdot K_{recipe}(u, v) \tag{2}$$

where $I$ is the Fourier transform of the intensity image $i$. (In general, we will use capital letters to denote functions in the Fourier domain). $K_{recipe}$ is a filter in the Fourier domain, of the same size as the image, whose construction is discussed below. Note that $K_{recipe}$ is zero in the low frequency bands where $Z_{low}$ is available. Once $z_{high}$ (the inverse Fourier transform of $Z_{high}$) is estimated, it can be combined with the known low-resolution range data simply by adding them together: $z_{recipe}(x, y) = z_{low}(x, y) + z_{high}(x, y)$.

For shorthand, we will write $I(u, v)I^*(u, v)$ as $II(u, v)$ and $Z(u, v)I^*(u, v)$ as $ZI(u, v)$. $II$ is also known as the power spectrum, and it is the Fourier transform of the autocorre-lation of the intensity image. $ZI$ is the Fourier transform of the cross-correlation between the intensity and range images, and it has both real and imaginary parts. Let $K = ZI/II$. Observe that $I \cdot K$ is a perfect reconstruction of the original high resolution range image (as long as $II(u, v) \neq 0$). Because we do not have the full-resolution range image, we can only compute the low spatial frequencies of $ZI(u, v)$. Let $K_{low} = ZI_{low}/II$, where $ZI_{low}$ is the Fourier transform of the cross-correlation between the low-resolution range image, and a low-resolution version of the intensity image. $K_{low}$ is zero in the high fre-quency bands. We can then think of $K_{recipe}$ as an approximation of $K = ZI/II$ formed by *extrapolating* $K_{low}$ into the higher spatial frequencies.

In the appendix, we show that shape recipes implicitly perform this extrapolation by learn-ing the highest available frequency octave of $K_{low}$, and duplicating this octave into all successive octaves of $K_{recipe}$, multiplied by a scale factor. However, there is a problem with this approach. First, there is no reason to expect that features in the range/intensity relationship should repeat once every octave. Figure 1a shows a plot of $ZI$ from a scene in our database of ground-truth range data (to be described in section 3). The fine structures in $\mathrm{real}[K]$ do not duplicate themselves every octave. Second and more importantly, octave duplication violates Freeman and Torralba's assumption that shape recipe kernels should change slowly over scale, which we take to mean over *all* scales, not just over successive octaves. Even if octave 2 of $K$ is made identical to octave 1, it is mathematically impossible for fractional octaves of $K$ like $1.5$ to also be identical unless $ZI/II$ is completely smooth and devoid of fine structure. The fine structures in $K$ therefore cannot possibly generalize over *all* scales.

In the next section, we use laser scans of real scenes to study the joint statistics of range and intensity images in greater detail, and use our results to form a statistically-motivated model of $ZI$. We believe that a greater understanding of the joint distribution of natural images and their underlying 3D structure will have a broad impact on the development of robust depth inference algorithms, and also on understanding human depth perception. More immediately, our statistical observations lead to a more accurate way to extrapolate $K_{low}$, which in turn results in a more accurate shape recipe method.

## 3 Scaling laws in natural scene statistics

To study the correlational structures between depth and intensity in natural scenes, we have collected a database of coregistered intensity and high-resolution range images (cor-responding pixels of the two images correspond to the same point in space). Scans were collected using the Riegl LMS-Z360 laser range scanner with integrated color photosensor.

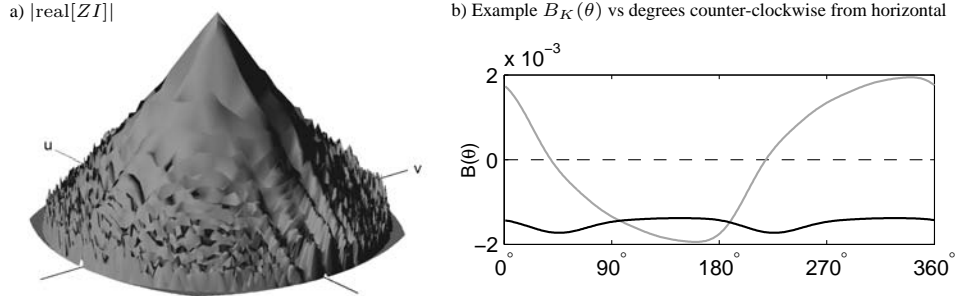

a) |real[$ZI$]|

b) Example $B_K(\theta)$ vs degrees counter-clockwise from horizontal

Figure 1: **a)** A log-log polar plot of |real[$ZI$]| from a scene in our database. $ZI$ contains extensive fine structures that do not repeat at each octave. However, along all orientations, the general form of |real[$ZI$]| is a power-law. |imag[$ZI$]| similarly obeys a power-law. **b)** A plot of $B_K(\theta)$ for the scene in figure 2. $real[B_K(\theta)]$ is drawn in black and $imag[B_K(\theta)]$ in grey. This plot is typical of most scenes in our database. As predicted by equation 4, $\mathrm{imag}[B_K(\theta)]$ reaches its minima at the illumination direction (in this case, to the extreme left, almost $180°$). Also typical is that $\mathrm{real}[B_K(\theta)]$ is uniformly negative, most likely caused by cast shadows in object concavities [6].

Scans were taken of a variety of rural and urban scenes. All images were taken outdoors, under sunny conditions, while the scanner was level with ground. The shape recipe model was intended for scenes with homogenous albedo and surface material. To test this algorithm in real scenes of this type, we selected 28 single-texture image sections from our database. These textures include statue surfaces and faceted building exteriors, such as archways and church facades (12 scenes), rocky terrain and rock piles (8), and leafy foliage (8). No logarithm or other transformation was applied to the intensity or range data (measured in meters), as this would interfere with the Lambertian model that motivates the shape recipe technique. Average size of these textures was 172,669 pixels per image.

We show a log-log polar plot of $|real[ZI(r,\theta)]|$ from one image in our database in figure 1a. As can be seen in the figure, this structure appears to closely follow a power law. We claim that $ZI$ can be reasonably modeled by $B(\theta)/r^\alpha$, where $r$ is spatial frequency in polar coordinates, and $B(\theta)$ is a parameter of the model (with both real and imaginary parts) that depends only on polar angle $\theta$. We test this claim by dividing the Fourier plane into four $45°$ octants (vertical, forward diagonal, horizontal, and backward diagonal), and measuring the drop-off rate in each octant separately. For each octant, we average over the octant's included orientations and fit the result to a power-law. The resulting values of $\alpha$ (averaged over all 28 images) are listed in the table below:

| orientation | $II$ | real[$ZI$] | imag[$ZI$] | $ZZ$ |
|---|---|---|---|---|
| horizontal | 2.47 ±0.10 | 3.61 ±0.18 | 3.84 ±0.19 | 2.84 ±0.11 |
| forward diagonal | 2.61 ±0.11 | 3.67 ±0.17 | 3.95 ±0.17 | 2.92 ±0.11 |
| vertical | 2.76 ±0.11 | 3.62 ±0.15 | 3.61 ±0.24 | 2.89 ±0.11 |
| backward diagonal | 2.56 ±0.09 | 3.69 ±0.17 | 3.84 ±0.23 | 2.86 ±0.10 |
| mean | 2.60 ±0.10 | 3.65 ±0.14 | 3.87 ±0.16 | 2.88 ±0.10 |

For each octant, the correlation coefficient between the power-law fit and the actual spectrum ranged from 0.91 to 0.99, demonstrating that each octant is well-fit by a power-law (Note that averaging over orientation smooths out some fine structures in each spectrum). Furthermore, $\alpha$ varies little across orientations, showing that our model fits $ZI$ closely.

The above findings predict that $K = ZI/II$ also obeys a power-law. Subtracting $\alpha_{II}$ from $\alpha_{\mathrm{real}[ZI]}$ and $\alpha_{\mathrm{imag}[ZI]}$, we find that $real[K]$ drops off at $1/r^{1.1}$ and $imag[K]$ drops off at $1/r^{1.2}$. Thus, we have that $K(r,\theta) \approx B_K(\theta)/r$.

Now that we know that $K$ can be fit (roughly) by a $1/r$ power-law, we can offer some insight into why $K$ tends to approximate this general form. The $1/r$ drop-off in the imaginary part of $K$ can be explained by the linear Lambertian model of shading, with oblique lighting conditions. This argument was used by Freeman and Torralba [9] in their theoretical motivation for choosing $c = 2$. The linear Lambertian model is obtained by taking only the linear terms of the Taylor series of the Lambertian equation. Under this model, if constant albedo is assumed, and no occlusion is present, then with lighting from above, $i(x, y) = a\,\partial z/\partial y$, where $a$ is some constant. In the Fourier domain, $I(u, v) = a2\pi jvZ(u, v)$, where $j = \sqrt{-1}$. Thus, we have that

$$ZI(r, \theta) = -\frac{j}{a2\pi\, r \sin(\theta)} II(r, \theta) \tag{3}$$

$$K(r, \theta) = -j\,\frac{1}{r}\,\frac{1}{a2\pi \sin(\theta)} \tag{4}$$

In other words, under this model, $K$ obeys a $1/r$ power-law. This means that each octave of $K$ is half of the octave before it. Our empirical finding that the imaginary part of $K$ obeys a $1/r$ power-law confirms Freeman and Torralba's reasoning behind choosing $c = 2$ for shape recipes.

However, the linear Lambertian shading model predicts that only the imaginary part of $ZI$ should obey a power-law. In fact, according to equation 3, this model predicts that the real part of $ZI$ should be zero. Yet, in our database, the real part of $ZI$ was typically stronger than the imaginary part. The real part of $ZI$ is the Fourier transform of the even-symmetric part of the cross-correlation function, and it includes the direct correlation $\mathrm{cov}[i, z]$. In a previous study of the statistics of natural range images [6], we have found that darker pixels in the image tend to be farther away, resulting in significantly negative $\mathrm{cov}[i, z]$. We attributed this phenomenon to cast shadows in complex scenes: object interiors and concavities are farther away than object exteriors, and these regions are the most likely to be in shadow. This effect can be observed wherever shadows are found, such as the crevices of figure 2a. However, the effect appears strongest in complex objects with many shadows and concavities, like folds of cloth, or foliage. We found that the real part of $ZI$ is especially likely to be strongly negative in images of foliage. Such correlation between depth and darkness has been predicted theoretically for diffuse lighting conditions, such as cloudy days, when viewed from directly above [5]. The fact that all of our images were taken under cloudless, sunny conditions and with oblique lighting from above suggests that this cue may be more important than at first realized. Psychophysical experiments have demonstrated that in the absence of all other cues, darker image regions appear farther, suggesting that the human visual system makes use of this cue for depth inference (see [6] for a review, also [10]). We believe that the $1/r$ drop-off rate observed in $\mathrm{real}[K]$ is due to the fact that concavities with smaller apertures but equal depths tend to be darker. In other words, for a given level of darkness, a smaller aperture corresponds to a more shallow hole.

## 4    Inference using power-law models

Armed with a better understanding of the statistics of real scenes, we are better prepared to develop successful depth inference algorithms. We now know that fine details in $ZI/II$ do not generalize across scales, but that its coarse structure roughly follows a $1/r$ power-law. We can exploit this statistical trend directly. We can simply fit our $B_K(\theta)/r$ power law to $ZI_{low}/II$, and then use this estimate of $K$ to reconstruct the high frequency range data.

Specifically, from the low-resolution range and intensity image, we compute low resolution spectra of $ZI$ and $II$. From the highest frequency octave of the low-resolution images, we estimate $B_{II}(\theta)$ and $B_{ZI}(\theta)$. Any standard interpolation method will work to estimate these functions. We chose a $cos^3(\theta + \pi\phi/4)$ basis function based on steerable filters [2].

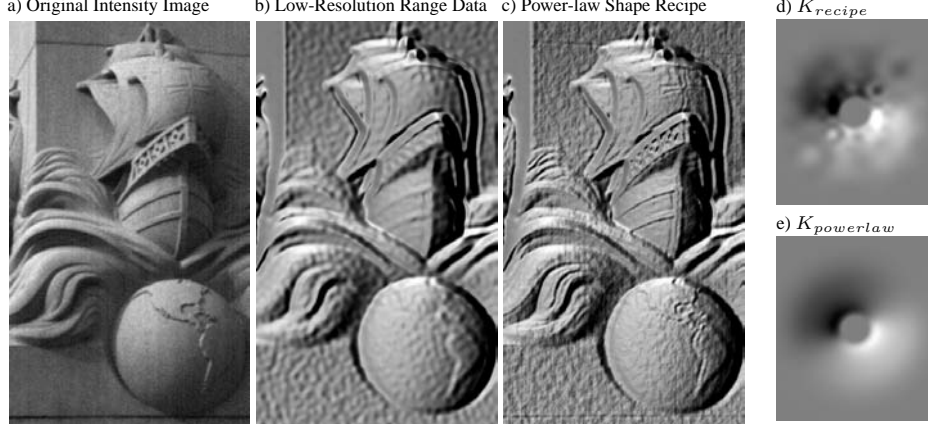

a) Original Intensity Image   b) Low-Resolution Range Data   c) Power-law Shape Recipe   d) $K_{recipe}$

e) $K_{powerlaw}$

Figure 2: **a)** An example intensity image from our database. **b)** A Lambertian rendering of the corresponding low resolution range image. **c)** Power-law method output. Shape recipe reconstructions show a similar amount of texture, but tests show that texture generated by the power-law method is more highly correlated with the true texture. **d)** The imaginary parts of $K_{recipe}$ and **e)** $K_{powerlaw}$ for the same scene. Dark regions are negative, light regions are positive. The grey center region in each estimate of $K$ corresponds to the low spatial frequencies, where range data is not inferred because it is already known. Notice that $K_{recipe}$ oscillates over scale.

We now can estimate the high spatial frequencies of the range image, $z$. Define

$$K_{powerlaw}(r, \theta) = F_{high}(r) \cdot (B_{ZI}(\theta)/B_{II}(\theta))/r \quad (5)$$

$$Z_{powerlaw} = Z_{low} + I \cdot K_{powerlaw} \quad (6)$$

where $F_{high}$ is the high-pass filter associated with the two highest resolution bands of the steerable filter pyramid of the full-resolution image.

## 5 Empirical evaluation

In this section, we compare the performance of shape recipes with our new approach, using our ground-truth database of high-resolution range and intensity image pairs described in section 3. For each range image in our database, a low-resolution (but still full-sized) range image, $z_{low}$, was generated by setting to zero the top two steerable filter pyramid layers. Both algorithms accepted as input the low-resolution range image and high-resolution intensity image, and the output was compared with the original high-resolution range image. The high resolution output corresponds to a 4-fold increase in spatial resolution (or a 16-fold increase in total size).

Although encouraging enhancements of stereo output were given by the authors, shape recipes has not been evaluated with real, ground-truth high resolution range data. To maximize its performance, we implemented shape recipes using ridge regression, with the ridge coefficient obtained using cross-validation. Linear kernels were learned (and the output evaluated) over a region of the image at least 21 pixels from the image border.

For each high-resolution output, we measured the sum squared error between the reconstruction ($z_{recipe}$ or $z_{powerlaw}$) and the original range image ($z$). We compared this with the sum-squared error of the low-resolution range image $z_{low}$ to get the percent reduction in sum-squared error: $error\_reduction_{recipe} = \frac{err_{low} - err_{recipe}}{err_{low}}$. This measure of error reflects the performance of the method independently of the variance or absolute depth of

the range image. On average, shape recipe reconstructions had $1.3\%$ less mean-squared error than $z_{low}$. Shape recipes improved 21 of the 28 images. Our new approach had $2.2\%$ less mean-squared error than $z_{low}$, and improved 26 of the 28 images.

We cannot expect the error reduction values to be very high, partly because our images are highly complex natural scenes, and also because some noise was present in both the range and intensity images. Therefore, it is difficult to assess how much of the remaining error could be recovered by a superior algorithm, and how much is simply due to sensor noise. As a comparison, we generated an optimal linear reconstruction, $z_{optlin}$, by learning $11 \times 11$ shape recipe kernels for the two high resolution pyramid bands directly from the ground-truth high resolution range image. This reconstruction provides a loose upper bound on the degree of improvement possible by linear shape methods. We then measured the percentage of linearly achievable improvement for each image: $improvement_{recipe} = \frac{err_{low} - err_{recipe}}{err_{low} - err_{optlin}}$ Shape recipes yielded an average improvement of $23\%$. Our approach achieved an improvement of $44\%$, nearly a two-fold enhancement over shape recipes.

## 6 The relative strengths of shading and shadow cues

Earlier we showed that Lambertian shading alone predicts that the real part of $ZI$ in natural scenes is empty of useful correlations between images and range images. Yet in our database, the real part of $ZI$, which we believe is related to shadow cues, was often *stronger* than the imaginary component. Our depth-inference algorithm offers an opportunity to compare the performance of shading cues versus shadow cues. We ran our algorithm again, except that we set the real part of $K_{powerlaw}$ to zero. This yielded only a $12\%$ improvement. However, when we ran the algorithm after setting $\mathrm{imag}[K]$ to zero, $32\%$ improvement was achieved. Thus, $72\%$ of the algorithm's total improvement was due to shadow cues. When the database is broken down into categories, the real part of $ZI$ is responsible for $96\%$ of total improvement in foliage scenes, $76\%$ in rocky terrain scenes, and $35\%$ in urban scenes (statue surfaces and building facades). As expected, the algorithm relies more heavily on the real part of $ZI$ in environments rich in cast shadows. These results show that shadow cues are far more useful than was previously expected, and also that they can be exploited more easily than was previously thought possible, using only simple linear relationships that might easily be incorporated into linear shape-from-shading techniques. We feel that these insights into natural scene statistics are the most important contributions of this paper.

## 7 Discussion

The power-law extension to shape recipes not only offers a substantial improvement in performance, but it also greatly reduces the number of parameters that must be learned. The original shape recipes required one $11 \times 11$ kernel, or 121 parameters, for each orientation of the steerable filters. The new algorithm requires only two parameters for each orientation (the real and the imaginary parts of $B_K(\theta)$). This suggests that the new approach has captured only those components of $K$ that generalize across scales, disregarding all others.

While it is encouraging that the power-law algorithm is highly parsimonious, it also means that fewer scene properties are encoded in the shape recipe kernels than was previously hoped [3]. For example, complex properties of the material and surface reflectance cannot be encoded. We believe that the $B(\theta)$ parameter of the power-law model can be determined almost entirely by the direction of illumination and the prominence of cast shadows (see figure 1b). This suggests that the power-law algorithm of this paper would work equally well for scenes with multiple materials. To capture more complex material properties, nonlinear methods and probabilistic methods may achieve greater success. However, when designing these more sophisticated methods, care must be taken to avoid the same pitfall encountered by shape recipes: not all properties of a scene can be scale-invariant simultaneously.

## 8 Appendix

Shape recipes infer each high resolution band of the range using equation 1. Let $\sigma = 2^{n-m}$. If we take the Fourier transform of equation 1, we get

$$Z_{high} \cdot F_{m,\phi} = \frac{1}{c^{n-m}} K_{n,\phi} \left( \frac{u}{\sigma}, \frac{v}{\sigma} \right) \cdot (I \cdot F_{m,\phi}) \tag{7}$$

where $F_{m,\phi}$ is the Fourier transform of the steerable filter at level $m$ and orientation $\phi$, and $Z_{high}$ is the inferred high spatial frequency components of the range image. If we take the steerable pyramid decomposition of $Z_{high}$ and then transform it back, we get $Z_{high}$ again, and so:

$$I \cdot K_{recipe} = Z_{high} = \sum_{m,\phi}^{m<n} Z_{high} F_{m,\phi} F_{m,\phi}^* \tag{8}$$

$$= I \sum_{m,\phi}^{m<n} \frac{1}{c^{n-m}} K_{n,\phi} \left( \frac{u}{\sigma}, \frac{v}{\sigma} \right) \cdot F_{m,\phi} \cdot F_{m,\phi}^* \tag{9}$$

The steerable filters at each level are simply a dilation of the steerable filters of preceding levels: $F_{m,\phi}(u,v) = F_{n,\phi} \left( \frac{u}{\sigma}, \frac{v}{\sigma} \right)$. Thus, recalling that $\sigma = 2^{n-m}$, we have

$$K_{recipe} = \sum_{m,\phi}^{m<n} \frac{1}{c^{n-m}} K_{n,\phi}(\frac{u}{\sigma}, \frac{v}{\sigma}) \cdot F_{n,\phi}(\frac{u}{\sigma}, \frac{v}{\sigma}) \cdot F_{n,\phi}^*(\frac{u}{\sigma}, \frac{v}{\sigma}) \tag{10}$$

The steerable filters $F_{n,\phi}$ are band-pass filters, and they are essentially zero outside of octave $n$. Thus, each octave of $K_{recipe}$ is identical to the octave before it, except reduced by a constant scale factor $c$. In other words, shape recipes extrapolate $K_{low}$ by copying the highest available octave of $K_{low}$ (or some estimation of it) into each successive octave. An example of $K_{recipe}$ can be seen in figure 2d.

*This research was funded in part by NSF IIS-0413211, Penn Dept of Health-MPC 05-06-2. Brian Potetz is supported by an NSF Graduate Research Fellowship.*

## References

[1] J. E. Cryer, P. S. Tsai and M. Shah, "Integration of shape from shading and stereo," Pattern Recognition, 28(7):1033–1043, 1995.

[2] W. T. Freeman, E. H. Adelson, "The design and use of steerable filters," IEEE Transactions on Pattern Analysis and Machine Intelligence, **13,** 891–906 1991.

[3] W. T. Freeman and A. Torralba, "Shape Recipes: Scene representations that refer to the image," Advances in Neural Information Processing Systems 15 (NIPS), MIT Press, 2003.

[4] C. Q. Howe and D. Purves, "Range image statistics can explain the anomalous perception of length," Proc. Nat. Acad. Sci. U.S.A. **99** 13184–13188 2002.

[5] M. S. Langer and S. W. Zucker, "Shape-from-shading on a cloudy day," J. Opt. Soc. Am. A **11,** 467–478 (1994).

[6] B. Potetz, T. S. Lee, "Statistical correlations between two-dimensional images and three-dimensional structures in natural scenes," J. Opt. Soc. Amer. A, **20,** 1292–1303 2003.

[7] D. Scharstein and R. Szeliski, "A taxonomy and evaluation of dense two-frame stereo correspondence algorithms," IJCV 47(1/2/3):7–42, April-June 2002.

[8] A. Torralba, A. Oliva, "Depth estimation from image structure," IEEE Transactions on Pattern Analysis and Machine Intelligence. 24(9): 1226–1238 2002.

[9] A. Torralba and W. T. Freeman, "Properties and applications of shape recipes," IEEE Computer Society Conference on Computer Vision and Pattern Recognition, 2003.

[10] C. W. Tyler, "Diffuse illumination as a default assumption for shape-from-shading in the absence of shadows," J. Imaging Sci. Technol. **42,** 319–325 1998.
